# Analysis and Improvement of Policy Gradient Estimation

**Tingting Zhao, Hirotaka Hachiya, Gang Niu, and Masashi Sugiyama**
Tokyo Institute of Technology
{tingting@sg., hachiya@sg., gang@sg., sugiyama@}cs.titech.ac.jp

## Abstract

*Policy gradient* is a useful model-free reinforcement learning approach, but it tends to suffer from instability of gradient estimates. In this paper, we analyze and improve the stability of policy gradient methods. We first prove that the variance of gradient estimates in the *PGPE* (policy gradients with parameter-based exploration) method is smaller than that of the classical REINFORCE method under a mild assumption. We then derive the optimal baseline for PGPE, which contributes to further reducing the variance. We also theoretically show that PGPE with the optimal baseline is more preferable than REINFORCE with the optimal baseline in terms of the variance of gradient estimates. Finally, we demonstrate the usefulness of the improved PGPE method through experiments.

## 1 Introduction

The goal of *reinforcement learning* (RL) is to find an optimal decision-making policy that maximizes the *return* (i.e., the sum of discounted rewards) through interaction with an unknown environment [13]. *Model-free* RL is a flexible framework in which decision-making policies are directly learned without going through explicit modeling of the environment. *Policy iteration* and *policy search* are two popular formulations of model-free RL.

In the policy iteration approach [6], the *value function* is first estimated and then policies are determined based on the learned value function. Policy iteration was demonstrated to work well in many real-world applications, especially in problems with discrete states and actions [14, 17, 1]. Although policy iteration can naturally deal with continuous states by function approximation [8], continuous actions are hard to handle due to the difficulty of finding maximizers of value functions with respect to actions. Moreover, since policies are indirectly determined via value function approximation, misspecification of value function models can lead to inappropriate policies even in very simple problems [15, 2]. Another limitation of policy iteration especially in physical control tasks is that control policies can vary drastically in each iteration. This causes severe instability in the physical system and thus is not favorable in practice.

Policy search is another approach to model-free RL that can overcome the limitations of policy iteration [18, 4, 7]. In the policy search approach, control policies are directly learned so that the return is maximized, for example, via a gradient method (called the *REINFORCE* method) [18], an EM algorithm [4], and a natural gradient method [7]. Among them, the gradient-based method is particularly useful in physical control tasks since policies are changed gradually. This ensures the stability of the physical system.

However, since the REINFORCE method tends to have a large variance in the estimation of the gradient directions, its naive implementation converges slowly [9, 10, 12]. Subtraction of the *optimal baseline* [16, 5] can ease this problem to some extent, but the variance of gradient estimates is still large. Furthermore, the performance heavily depends on the choice of an initial policy, and appropriate initialization is not straightforward in practice.

To cope with this problem, a novel policy gradient method called *policy gradients with parameter-based exploration* (PGPE) was proposed recently [12]. In PGPE, an initial policy is drawn from a prior probability distribution, and then actions are chosen deterministically. This construction contributes to mitigating the problem of initial policy choice and stabilizing gradient estimates. Moreover, by subtracting a moving-average baseline, the variance of gradient estimates can be further reduced. Through robot-control experiments, PGPE was demonstrated to achieve more stable performance than existing policy-gradient methods.

The goal of this paper is to theoretically support the usefulness of PGPE, and to further improve its performance. More specifically, we first give bounds of the gradient estimates of the REINFORCE and PGPE methods. Our theoretical analysis shows that gradient estimates for PGPE have smaller variance than those for REINFORCE under a mild condition. We then show that the moving-average baseline for PGPE adopted in the original paper [12] has excess variance; we give the optimal baseline for PGPE that minimizes the variance, following the line of [16, 5]. We further theoretically show that PGPE with the optimal baseline is more preferable than REINFORCE with the optimal baseline in terms of the variance of gradient estimates. Finally, the usefulness of the improved PGPE method is demonstrated through experiments.

## 2 Policy Gradients for Reinforcement Learning

In this section, we review policy gradient methods.

### 2.1 Problem Formulation

Let us consider a Markov decision problem specified by $(\mathcal{S}, \mathcal{A}, P_T, P_I, r, \gamma)$, where $\mathcal{S}$ is a set of $\ell$-dimensional continuous states, $\mathcal{A}$ is a set of continuous actions, $P_T(\boldsymbol{s}'|\boldsymbol{s}, a)$ is the transition probability density from current state $\boldsymbol{s}$ to next state $\boldsymbol{s}'$ when action $a$ is taken, $P_I(\boldsymbol{s})$ is the probability of initial states, $r(\boldsymbol{s}, a, \boldsymbol{s}')$ is an immediate reward for transition from $\boldsymbol{s}$ to $\boldsymbol{s}'$ by taking action $a$, and $0 < \gamma < 1$ is the discount factor for future rewards. Let $p(a|\boldsymbol{s}, \boldsymbol{\theta})$ be a stochastic policy with parameter $\boldsymbol{\theta}$, which represents the conditional probability density of taking action $a$ in state $\boldsymbol{s}$.

Let $h = [\boldsymbol{s}_1, a_1, \ldots, \boldsymbol{s}_T, a_T]$ be a *trajectory* of length $T$. Then the *return* (i.e., the discounted sum of future rewards) along $h$ is given by

$$R(h) := \sum_{t=1}^{T} \gamma^{t-1} r(\boldsymbol{s}_t, a_t, \boldsymbol{s}_{t+1}).$$

The expected return for parameter $\boldsymbol{\theta}$ is defined by

$$J(\boldsymbol{\theta}) := \int p(h|\boldsymbol{\theta}) R(h) \mathrm{d}h, \quad \text{where} \quad p(h|\boldsymbol{\theta}) = p(\boldsymbol{s}_1) \prod_{t=1}^{T} p(\boldsymbol{s}_{t+1}|\boldsymbol{s}_t, a_t) p(a_t|\boldsymbol{s}_t, \boldsymbol{\theta}).$$

The goal of reinforcement learning is to find the optimal policy parameter $\boldsymbol{\theta}^*$ that maximizes the expected return $J(\boldsymbol{\theta})$:

$$\boldsymbol{\theta}^* := \arg\max J(\boldsymbol{\theta}).$$

### 2.2 Review of the REINFORCE Algorithm

In the *REINFORCE* algorithm [18], the policy parameter $\boldsymbol{\theta}$ is updated via *gradient ascent*:

$$\boldsymbol{\theta} \longleftarrow \boldsymbol{\theta} + \varepsilon \nabla_{\boldsymbol{\theta}} J(\boldsymbol{\theta}),$$

where $\varepsilon$ is a small positive constant. The gradient $\nabla_{\boldsymbol{\theta}} J(\boldsymbol{\theta})$ is given by

$$\nabla_{\boldsymbol{\theta}} J(\boldsymbol{\theta}) = \int \nabla_{\boldsymbol{\theta}} p(h|\boldsymbol{\theta}) R(h) \mathrm{d}h = \int p(h|\boldsymbol{\theta}) \nabla_{\boldsymbol{\theta}} \log p(h|\boldsymbol{\theta}) R(h) \mathrm{d}h$$

$$= \int p(h|\boldsymbol{\theta}) \sum_{t=1}^{T} \nabla_{\boldsymbol{\theta}} \log p(a_t|\boldsymbol{s}_t, \boldsymbol{\theta}) R(h) \mathrm{d}h,$$

where we used the so-called 'log trick': $\nabla_{\boldsymbol{\theta}} p(h|\boldsymbol{\theta}) = p(h|\boldsymbol{\theta}) \nabla_{\boldsymbol{\theta}} \log p(h|\boldsymbol{\theta})$. Since $p(h|\boldsymbol{\theta})$ is unknown, the expectation is approximated by the empirical average:

$$\nabla_{\boldsymbol{\theta}} \widehat{J}(\boldsymbol{\theta}) = \frac{1}{N} \sum_{n=1}^{N} \sum_{t=1}^{T} \nabla_{\boldsymbol{\theta}} \log p(a_t^n|\boldsymbol{s}_t^n, \boldsymbol{\theta}) R(h^n),$$

where $h^n := [\boldsymbol{s}_1^n, a_1^n, \ldots, \boldsymbol{s}_T^n, a_T^n]$ is a roll-out sample.

Let us employ the Gaussian policy model with parameter $\boldsymbol{\theta} = (\boldsymbol{\mu}, \sigma)$, where $\boldsymbol{\mu}$ is the mean vector and $\sigma$ is the standard deviation:

$$p(a|\boldsymbol{s}, \boldsymbol{\theta}) = \frac{1}{\sigma\sqrt{2\pi}} \exp\left(-\frac{(a-\boldsymbol{\mu}^\top\boldsymbol{s})^2}{2\sigma^2}\right).$$

Then the policy gradients are explicitly given as

$$\nabla_{\boldsymbol{\mu}} \log p(a|\boldsymbol{s}, \boldsymbol{\theta}) = \frac{a-\boldsymbol{\mu}^\top\boldsymbol{s}}{\sigma^2}\boldsymbol{s} \quad \text{and} \quad \nabla_\sigma \log p(a|\boldsymbol{s}, \boldsymbol{\theta}) = \frac{(a-\boldsymbol{\mu}^\top\boldsymbol{s})^2 - \sigma^2}{\sigma^3}.$$

A drawback of REINFORCE is that the variance of the above policy gradients is large [10, 11], which leads to slow convergence.

## 2.3 Review of the PGPE Algorithm

One of the reasons for large variance of policy gradients in the REINFORCE algorithm is that the empirical average is taken at each time step, which is caused by stochasticity of policies.

In order to mitigate this problem, another method called *policy gradients with parameter-based exploration* (PGPE) was proposed recently [11]. In PGPE, a linear *deterministic* policy,

$$\pi(a|\boldsymbol{s}, \boldsymbol{\theta}) = \boldsymbol{\theta}^\top\boldsymbol{s},$$

is adopted, and stochasticity is introduced by considering a prior distribution over policy parameter $\boldsymbol{\theta}$ with hyper-parameter $\boldsymbol{\rho}$: $p(\boldsymbol{\theta}|\boldsymbol{\rho})$. Since entire history $h$ is solely determined by a single sample of parameter $\boldsymbol{\theta}$ in this formulation, it is expected that the variance of gradient estimates can be reduced.

The expected return for hyper-parameter $\boldsymbol{\rho}$ is expressed as

$$J(\boldsymbol{\rho}) = \iint p(h|\boldsymbol{\theta})p(\boldsymbol{\theta}|\boldsymbol{\rho})R(h)\mathrm{d}h\mathrm{d}\boldsymbol{\theta}.$$

Differentiating this with respect to $\boldsymbol{\rho}$, we have

$$\nabla_{\boldsymbol{\rho}}J(\boldsymbol{\rho}) = \iint p(h|\boldsymbol{\theta})\nabla_{\boldsymbol{\rho}}p(\boldsymbol{\theta}|\boldsymbol{\rho})R(h)\mathrm{d}h\mathrm{d}\boldsymbol{\theta} = \iint p(h|\boldsymbol{\theta})p(\boldsymbol{\theta}|\boldsymbol{\rho})\nabla_{\boldsymbol{\rho}} \log p(\boldsymbol{\theta}|\boldsymbol{\rho})R(h)\mathrm{d}h\mathrm{d}\boldsymbol{\theta},$$

where the log trick for $\nabla_{\boldsymbol{\rho}}p(\boldsymbol{\theta}|\boldsymbol{\rho})$ is used. We then approximate the expectation over $h$ and $\boldsymbol{\theta}$ by the empirical average:

$$\nabla_{\boldsymbol{\rho}}\widehat{J}(\boldsymbol{\rho}) = \frac{1}{N} \sum_{n=1}^{N} \nabla_{\boldsymbol{\rho}} \log p(\boldsymbol{\theta}^n|\boldsymbol{\rho})R(h^n),$$

where each trajectory sample $h^n$ is drawn from $p(h|\boldsymbol{\theta}^n)$ and the parameter $\boldsymbol{\theta}^n$ is drawn from $p(\boldsymbol{\theta}^n|\boldsymbol{\rho})$.

Let us employ the Gaussian prior distribution with hyper-parameter $\boldsymbol{\rho} = (\boldsymbol{\eta}, \boldsymbol{\tau})$ to draw parameter vector $\boldsymbol{\theta}$, where $\boldsymbol{\eta}$ is the mean vector and $\boldsymbol{\tau}$ is the vector consisting of the standard deviation in each element:

$$p(\theta_i|\boldsymbol{\rho}_i) = \frac{1}{\tau_i\sqrt{2\pi}} \exp\left(-\frac{(\theta_i-\eta_i)^2}{2\tau_i^2}\right).$$

Then the derivative of $\log p(\boldsymbol{\theta}|\boldsymbol{\rho})$ with respect to $\eta_i$ and $\tau_i$ are given as follows:

$$\nabla_{\eta_i} \log p(\boldsymbol{\theta}|\boldsymbol{\rho}) = \frac{\theta_i-\eta_i}{\tau_i^2} \quad \text{and} \quad \nabla_{\tau_i} \log p(\boldsymbol{\theta}|\boldsymbol{\rho}) = \frac{(\theta_i - \eta_i)^2 - \tau_i^2}{\tau_i^3}.$$

# 3 Variance of Gradient Estimates

In this section, we theoretically investigate the variance of gradient estimates in REINFORCE and PGPE.

For multi-dimensional state space, we consider the *trace* of the covariance matrix of gradient vectors. That is, for a random vector $\boldsymbol{A} = (A_1, \ldots, A_\ell)^\top$, we define

$$\mathbf{Var}(\boldsymbol{A}) = \mathrm{tr}\left(\mathbb{E}\left[(\boldsymbol{A} - \mathbb{E}[\boldsymbol{A}])(\boldsymbol{A} - \mathbb{E}[\boldsymbol{A}])^\top\right]\right) = \sum_{m=1}^{\ell} \mathbb{E}\left[(A_m - \mathbb{E}[A_m])^2\right], \qquad (1)$$

where $\mathbb{E}$ denotes the expectation. Let $B = \sum_{i=1}^{\ell} \tau_i^{-2}$, where $\ell$ is the dimensionality of state $\boldsymbol{s}$.

Below, we consider a subset of the following assumptions:

**Assumption (A):** $r(\boldsymbol{s}, a, \boldsymbol{s}') \in [-\beta, \beta]$ for $\beta > 0$.

**Assumption (B):** $r(\boldsymbol{s}, a, \boldsymbol{s}') \in [\alpha, \beta]$ for $0 < \alpha < \beta$.

**Assumption (C):** For $\delta > 0$, there exist two series $\{c_t\}_{t=1}^{T}$ and $\{d_t\}_{t=1}^{T}$ such that $\|\boldsymbol{s}_t\|_2 \geq c_t$ and $\|\boldsymbol{s}_t\|_2 \leq d_t$ hold with probability at least $(1-\delta)^{1/2N}$ respectively over the choice of sample paths, where $\|\cdot\|_2$ denotes the $\ell_2$-norm.

Note that Assumption (B) is stronger than Assumption (A). Let

$$\mathcal{L}(T) = C_T \alpha^2 - D_T \beta^2/(2\pi), \quad C_T = \sum_{t=1}^{T} c_t^2, \quad \text{and} \quad D_T = \sum_{t=1}^{T} d_t^2.$$

First, we analyze the variance of gradient estimates in PGPE (the proofs of all the theorems are provided in the supplementary material):

**Theorem 1.** *Under Assumption (A), we have the following upper bounds:*

$$\mathbf{Var}\left[\nabla_{\boldsymbol{\eta}} \widehat{J}(\boldsymbol{\rho})\right] \leq \frac{\beta^2 (1-\gamma^T)^2 B}{N(1-\gamma)^2} \quad \text{and} \quad \mathbf{Var}\left[\nabla_{\boldsymbol{\tau}} \widehat{J}(\boldsymbol{\rho})\right] \leq \frac{2\beta^2 (1-\gamma^T)^2 B}{N(1-\gamma)^2},$$

This theorem means that the upper bound of the variance of $\nabla_{\boldsymbol{\eta}} \widehat{J}(\boldsymbol{\rho})$ is proportional to $\beta^2$ (the upper bound of squared rewards), $B$ (the trace of the inverse Gaussian covariance), and $(1-\gamma^T)^2/(1-\gamma)^2$, and is inverse-proportional to sample size $N$. The upper bound of the variance of $\nabla_{\boldsymbol{\tau}} \widehat{J}(\boldsymbol{\rho})$ is twice larger than that of $\nabla_{\boldsymbol{\eta}} \widehat{J}(\boldsymbol{\rho})$. When $T$ goes to infinity, $(1 - \gamma^T)^2$ will converge to 1.

Next, we analyze the variance of gradient estimates in REINFORCE:

**Theorem 2.** *Under Assumptions (B) and (C), we have the following lower bound with probability at least $1 - \delta$:*

$$\mathbf{Var}\left[\nabla_{\boldsymbol{\mu}} \widehat{J}(\boldsymbol{\theta})\right] \geq \frac{(1-\gamma^T)^2}{N\sigma^2(1-\gamma)^2} \mathcal{L}(T).$$

*Under Assumptions (A) and (C), we have the following upper bound with probability at least $(1 - \delta)^{1/2}$:*

$$\mathbf{Var}\left[\nabla_{\boldsymbol{\mu}} \widehat{J}(\boldsymbol{\theta})\right] \leq \frac{D_T \beta^2 (1-\gamma^T)^2}{N\sigma^2(1-\gamma)^2}.$$

*Under Assumption (A), we have*

$$\mathbf{Var}\left[\nabla_{\sigma} \widehat{J}(\boldsymbol{\theta})\right] \leq \frac{2T\beta^2 (1-\gamma^T)^2}{N\sigma^2(1-\gamma)^2}.$$

The upper bounds for REINFORCE are similar to those for PGPE, but they are monotone increasing with respect to trajectory length $T$. The lower bound for the variance of $\nabla_{\boldsymbol{\mu}} \widehat{J}(\boldsymbol{\theta})$ will be non-trivial if it is positive, i.e., $\mathcal{L}(T) > 0$. This can be fulfilled, e.g., if $\alpha$ and $\beta$ satisfy $2\pi C_T \alpha^2 > D_T \beta^2$. Deriving a lower bound of the variance of $\nabla_{\sigma} \widehat{J}(\boldsymbol{\theta})$ is left open as future work.

Finally, we compare the variance of gradient estimates in REINFORCE and PGPE:

**Theorem 3.** *In addition to Assumptions (B) and (C), we assume $\mathcal{L}(T)$ is positive and monotone increasing with respect to $T$. If there exists $T_0$ such that $\mathcal{L}(T_0) \geq \beta^2 B \sigma^2$, then we have $\mathbf{Var}[\nabla_{\boldsymbol{\mu}} \widehat{J}(\boldsymbol{\theta})] > \mathbf{Var}[\nabla_{\boldsymbol{\eta}} \widehat{J}(\boldsymbol{\rho})]$ for all $T > T_0$, with probability at least $1 - \delta$.*

The above theorem means that PGPE is more favorable than REINFORCE in terms of the variance of gradient estimates of the mean, if trajectory length $T$ is large. This theoretical result would partially support the experimental success of the PGPE method [12].

## 4 Variance Reduction by Subtracting Baseline

In this section, we give a method to reduce the variance of gradient estimates in PGPE and analyze its theoretical properties.

## 4.1 Basic Idea of Introducing Baseline

It is known that the variance of gradient estimates can be reduced by subtracting a *baseline* $b$: for REINFORCE and PGPE, modified gradient estimates are given by

$$\nabla_{\boldsymbol{\theta}} \widehat{J}^b(\boldsymbol{\theta}) = \frac{1}{N} \sum_{n=1}^{N} (R(h^n) - b) \sum_{t=1}^{T} \nabla_{\boldsymbol{\theta}} \log p(a_t^n | \boldsymbol{s}_t^n, \boldsymbol{\theta}),$$

$$\nabla_{\boldsymbol{\rho}} \widehat{J}^b(\boldsymbol{\rho}) = \frac{1}{N} \sum_{n=1}^{N} (R(h^n) - b) \nabla_{\boldsymbol{\rho}} \log p(\boldsymbol{\theta}^n | \boldsymbol{\rho}).$$

The *adaptive reinforcement baseline* [18] was derived as the exponential moving average of the past experience:

$$b(n) = \gamma R(h^{n-1}) + (1 - \gamma) b(n - 1),$$

where $0 < \gamma \leq 1$. Based on this, an empirical gradient estimate with the moving-average baseline was proposed for REINFORCE [18] and PGPE [12].

The above moving-average baseline contributes to reducing the variance of gradient estimates. However, it was shown [5, 16] that the moving-average baseline is not optimal; the optimal baseline is, by definition, given as the minimizer of the variance of gradient estimates with respect to a baseline. Following this formulation, the optimal baseline for REINFORCE is given as follows [10]:

$$b^*_{\text{REINFORCE}} := \arg\min_b \mathbf{Var}[\nabla_{\boldsymbol{\theta}} \widehat{J}^b(\boldsymbol{\theta})] = \frac{\mathbb{E}[R(h) \| \sum_{t=1}^{T} \nabla_{\boldsymbol{\theta}} \log p(a_t | \boldsymbol{s}_t, \boldsymbol{\theta}) \|^2]}{\mathbb{E}[\| \sum_{t=1}^{T} \nabla_{\boldsymbol{\theta}} \log p(a_t | \boldsymbol{s}_t, \boldsymbol{\theta}) \|^2]}.$$

However, only the moving-average baseline was introduced to PGPE so far [12], which is suboptimal. Below, we derive the optimal baseline for PGPE, and study its theoretical properties.

## 4.2 Optimal Baseline for PGPE

Let $b^*_{\text{PGPE}}$ be the optimal baseline for PGPE that minimizes the variance:

$$b^*_{\text{PGPE}} := \arg\min_b \mathbf{Var}[\nabla_{\boldsymbol{\rho}} \widehat{J}^b(\boldsymbol{\rho})].$$

Then the following theorem gives the optimal baseline for PGPE:

**Theorem 4.** *The optimal baseline for PGPE is given by*

$$b^*_{\text{PGPE}} = \frac{\mathbb{E}[R(h) \| \nabla_{\boldsymbol{\rho}} \log p(\boldsymbol{\theta} | \boldsymbol{\rho}) \|^2]}{\mathbb{E}[\| \nabla_{\boldsymbol{\rho}} \log p(\boldsymbol{\theta} | \boldsymbol{\rho}) \|^2]},$$

*and the excess variance for a baseline $b$ is given by*

$$\mathbf{Var}[\nabla_{\boldsymbol{\rho}} \widehat{J}^b(\boldsymbol{\rho})] - \mathbf{Var}[\nabla_{\boldsymbol{\rho}} \widehat{J}^{b^*_{\text{PGPE}}}(\boldsymbol{\rho})] = \frac{(b - b^*_{\text{PGPE}})^2}{N} \mathbb{E}[\| \nabla_{\boldsymbol{\rho}} \log p(\boldsymbol{\theta} | \boldsymbol{\rho}) \|^2].$$

The above theorem gives an analytic-form expression of the optimal baseline for PGPE. When expected return $R(h)$ and the squared norm of characteristic eligibility $\| \nabla_{\boldsymbol{\rho}} \log p(\boldsymbol{\theta} | \boldsymbol{\rho}) \|^2$ are independent of each other, the optimal baseline is reduced to average expected return $\mathbb{E}[R(h)]$. However, the optimal baseline is generally different from the average expected return. The above theorem also shows that the excess variance is proportional to the squared difference of baselines $(b - b^*_{\text{PGPE}})^2$ and the expected squared norm of characteristic eligibility $\mathbb{E}[\| \nabla_{\boldsymbol{\rho}} \log p(\boldsymbol{\theta} | \boldsymbol{\rho}) \|^2]$, and is inverse-proportional to sample size $N$.

Next, we analyze the contribution of the optimal baseline to the variance with respect to mean parameter $\boldsymbol{\eta}$ in PGPE:

**Theorem 5.** *If $r(\boldsymbol{s}, a, \boldsymbol{s}') \geq \alpha > 0$, we have the following lower bound:*

$$\mathbf{Var}[\nabla_{\boldsymbol{\eta}} \widehat{J}(\boldsymbol{\rho})] - \mathbf{Var}[\nabla_{\boldsymbol{\eta}} \widehat{J}^{b^*_{\text{PGPE}}}(\boldsymbol{\rho})] \geq \frac{\alpha^2 (1 - \gamma^T)^2 B}{N (1 - \gamma)^2}.$$

*Under Assumption (A), we have the following upper bound:*

$$\mathbf{Var}[\nabla_{\boldsymbol{\eta}} \widehat{J}(\boldsymbol{\rho})] - \mathbf{Var}[\nabla_{\boldsymbol{\eta}} \widehat{J}^{b^*_{\text{PGPE}}}(\boldsymbol{\rho})] \leq \frac{\beta^2 (1 - \gamma^T)^2 B}{N (1 - \gamma)^2}.$$

This theorem shows that the lower and upper bounds of the excess variance are proportional to $\alpha^2$ and $\beta^2$ (the bounds of squared immediate rewards), $B$ (the trace of the inverse Gaussian covariance), and $(1 - \gamma^T)^2 / (1 - \gamma)^2$, and are inverse-proportional to sample size $N$. When $T$ goes to infinity, $(1 - \gamma^T)^2$ will converge to 1.

## 4.3 Comparison with REINFORCE

Next, we analyze the contribution of the optimal baseline for REINFORCE, and compare it with that for PGPE. It was shown [5, 16] that the excess variance for a baseline $b$ in REINFORCE is given by

$$\mathbf{Var}[\nabla_{\boldsymbol{\theta}}\widehat{J}^b(\boldsymbol{\theta})] - \mathbf{Var}[\nabla_{\boldsymbol{\theta}}\widehat{J}^{b^*_{\text{REINFORCE}}}(\boldsymbol{\theta})] = \frac{(b-b^*_{\text{REINFORCE}})^2}{N}\mathbb{E}\left[\left\|\sum_{t=1}^{T}\nabla_{\boldsymbol{\theta}}\log p(a_t|\boldsymbol{s}_t,\boldsymbol{\theta})\right\|^2\right].$$

Based on this, we have the following theorem:

**Theorem 6.** *Under Assumptions (B) and (C), we have the following bounds with probability at least $1 - \delta$:*

$$\frac{C_T\alpha^2(1-\gamma^T)^2}{N\sigma^2(1-\gamma)^2} \leq \mathbf{Var}[\nabla_{\boldsymbol{\mu}}\widehat{J}(\boldsymbol{\theta})] - \mathbf{Var}[\nabla_{\boldsymbol{\mu}}\widehat{J}^{b^*_{\text{REINFORCE}}}(\boldsymbol{\theta})] \leq \frac{\beta^2(1-\gamma^T)^2 D_T}{N\sigma^2(1-\gamma)^2}.$$

The above theorem shows that the lower and upper bounds of the excess variance are monotone increasing with respect to trajectory length $T$.

In the aspect of the amount of reduction in the variance of gradient estimates, Theorem 5 and Theorem 6 show that the optimal baseline for REINFORCE contributes more than that for PGPE.

Finally, based on Theorem 1 and Theorem 5 and based on Theorem 2 and Theorem 6, we have the following theorem:

**Theorem 7.** *Under Assumptions (B) and (C), we have*

$$\mathbf{Var}[\nabla_{\boldsymbol{\eta}}\widehat{J}^{b^*_{\text{PGPE}}}(\boldsymbol{\rho})] \leq \frac{(1-\gamma^T)^2}{N(1-\gamma)^2}(\beta^2 - \alpha^2)B,$$

$$\mathbf{Var}[\nabla_{\boldsymbol{\mu}}\widehat{J}^{b^*_{\text{REINFORCE}}}(\boldsymbol{\theta})] \leq \frac{(1-\gamma^T)^2}{N\sigma^2(1-\gamma)^2}(\beta^2 D_T - \alpha^2 C_T),$$

*where the latter inequality holds with probability at least $1 - \delta$.*

This theorem shows that the upper bound of the variance of gradient estimates for REINFORCE with the optimal baseline is still monotone increasing with respect to trajectory length $T$. On the other hand, since $(1 - \gamma^T)^2 \leq 1$, the above upper bound of the variance of gradient estimates in PGPE with the optimal baseline can be further upper-bounded as $\mathbf{Var}[\nabla_{\boldsymbol{\eta}}\widehat{J}^{b^*_{\text{PGPE}}}(\boldsymbol{\rho})] \leq \frac{(\beta^2-\alpha^2)B}{N(1-\gamma)^2}$, which is independent of $T$. Thus, when trajectory length $T$ is large, the variance of gradient estimates in REINFORCE with the optimal baseline may be significantly larger than the variance of gradient estimates in PGPE with the optimal baseline.

## 5 Experiments

In this section, we experimentally investigate the usefulness of the proposed method, PGPE with the optimal baseline.

### 5.1 Illustrative Data

Let the state space $\mathcal{S}$ be one-dimensional and continuous, and the initial state is randomly chosen from the standard normal distribution. The action space $\mathcal{A}$ is also set to be one-dimensional and continuous. The transition dynamics of the environment is set at $s_{t+1} = s_t + a_t + \varepsilon$, where $\varepsilon \sim \mathcal{N}(0, 0.5^2)$ is stochastic noise. The immediate reward is defined as $r = \exp\left(-s^2/2 - a^2/2\right) + 1$, which is bounded as $1 < r \leq 2$. The discount factor is set at $\gamma = 0.9$.

Here, we compare the following five methods: REINFORCE without any baselines, REINFORCE with the optimal baseline (OB), PGPE without any baselines, PGPE with the moving-average baseline (MB), and PGPE with the optimal baseline (OB). For fair comparison, all of these methods use the same parameter setup: the mean and standard deviation of the Gaussian distribution is set at $\mu = -1.5$ and $\sigma = 1$, the number of episodic samples is set at $N = 100$, and the length of the trajectory is set at $T = 10$ or $50$. We then calculate the variance of gradient estimates over 100 runs.

Table 1 summarizes the results, showing that the variance of REINFORCE is overall larger than PGPE. A notable difference between REINFORCE and PGPE is that the variance of REINFORCE

Table 1: Variance and bias of estimated gradients for the illustrative data.

| Method | $T = 10$ | | | | $T = 50$ | | | |
| | Variance | | Bias | | Variance | | Bias | |
| | $\mu, \eta$ | $\sigma, \tau$ | $\mu, \eta$ | $\sigma, \tau$ | $\mu, \eta$ | $\sigma, \tau$ | $\mu, \eta$ | $\sigma, \tau$ |
|---|---|---|---|---|---|---|---|---|
| REINFORCE | 13.2570 | 26.9173 | -0.3102 | -1.5098 | 188.3860 | 278.3095 | -1.8126 | -5.1747 |
| REINFORCE-OB | 0.0914 | 0.1203 | 0.0672 | 0.1286 | 0.5454 | 0.8996 | -0.2988 | -0.2008 |
| PGPE | 0.9707 | 1.6855 | -0.0691 | 0.1319 | 1.6572 | 3.3720 | -0.1048 | -0.3293 |
| PGPE-MB | 0.2127 | 0.3238 | 0.0828 | -0.1295 | 0.4123 | 0.8332 | 0.0925 | -0.2556 |
| PGPE-OB | 0.0372 | 0.0685 | -0.0164 | 0.0512 | 0.0850 | 0.1815 | 0.0480 | -0.0779 |

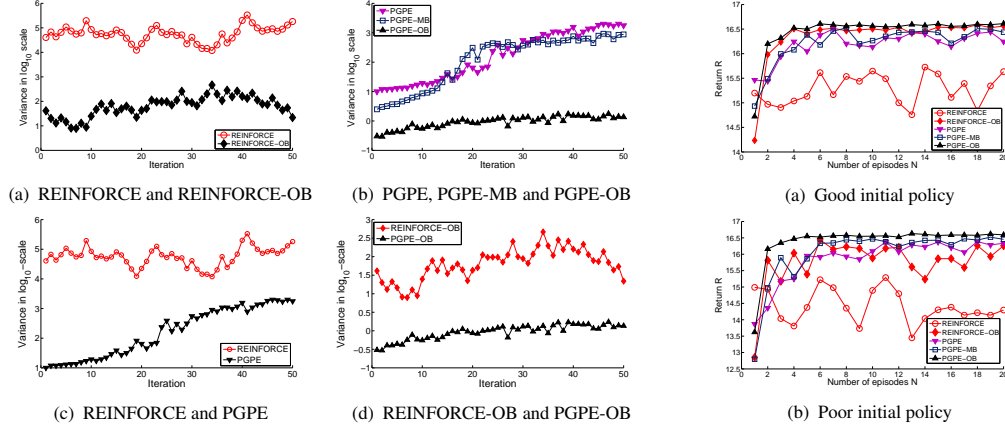

(a) REINFORCE and REINFORCE-OB

(b) PGPE, PGPE-MB and PGPE-OB

(a) Good initial policy

(c) REINFORCE and PGPE

(d) REINFORCE-OB and PGPE-OB

(b) Poor initial policy

Figure 1: Variance of gradient estimates with respect to the mean parameter through policy-update iterations for the illustrative data.

Figure 2: Return as functions of the number of episodic samples $N$ for the illustrative data.

significantly grows as $T$ increases, whereas that of PGPE is not influenced that much by $T$. This well agrees with our theoretical analysis in Section 3. The results also show that the variance of PGPE-OB (the proposed method) is much smaller than that of PGPE-MB. REINFORCE-OB contributes highly to reducing the variance especially when $T$ is large, which also well agrees with our theory. However, PGPE-OB still provides much smaller variance than REINFORCE-OB.

We also investigate the bias of gradient estimates of each method. We regard gradients estimated with $N = 1000$ as true gradients, and compute the bias of gradient estimates when $N = 100$. The results are also included in Table 1, showing that introduction of baselines does not increase the bias; rather, it tends to reduce the bias.

Next, we investigate the variance of gradient estimates when policy parameters are updated over iterations. In this experiment, we set $N = 10$ and $T = 20$, and the variance is computed from 50 runs. Policies are updated over 50 iterations. In order to evaluate the variance in a stable manner, we repeat the above experiments 20 times with random choice of initial mean parameter $\mu$ from $[-3.0, -0.1]$, and investigate the average variance of gradient estimates with respect to mean parameter $\mu$ over 20 trials, in $\log_{10}$-scale.

The results are summarized in Figure 1. Figure 1(a) compares the variance of REINFORCE with/without baselines, whereas Figure 1(b) compares the variance of PGPE with/without baselines. These plots show that introduction of baselines contributes highly to the reduction of the variance over iterations. Figure 1(c) compares the variance of REINFORCE and PGPE without baselines, showing that PGPE provides much more stable gradient estimates than REINFORCE. Figure 1(d) compares the variance of REINFORCE and PGPE with the optimal baselines, showing that gradient estimates obtained by PGPE-OB are much smaller than those by REINFORCE-OB. Overall, in terms of the variance of gradient estimates, the proposed PGPE-OB compares favorably with other methods.

Next, we evaluate returns obtained by each method. The trajectory length is fixed at $T = 20$, and the maximum number of policy-update iterations is set at 50. We investigate average returns over 20 runs as functions of the number of episodic samples $N$. We have two experimental results for different initial policies. Figure 2(a) shows the results when initial mean parameter $\mu$ is chosen randomly

from $[-1.6, -0.1]$, which tends to perform well. The graph shows that PGPE-OB performs the best, especially when $N < 5$; then REINFORCE-OB follows with a small margin. PGPE-MB and plain PGPE also work reasonably well, although they are slightly unstable due to larger variance. Plain REINFORCE is highly unstable, which is caused by the huge variance of gradient estimates (see Figure 1 again).

Figure 2(b) describes the results when initial mean parameter $\mu$ is chosen randomly from $[-3.0, -0.1]$, which tends to result in poorer performance. In this setup, difference among the compared methods is more significant than the case with good initial policies. Overall, plain REINFORCE performs very poorly, and even REINFORCE-OB tends to be outperformed by the PGPE methods. This means that REINFORCE is very sensitive to the choice of initial policies. Among the PGPE methods, the proposed PGPE-OB works very well and converges quickly.

## 5.2 Cart-Pole Balancing

Finally, we evaluate the performance of our proposed method in a more complex task of *cart-pole balancing* [3]. A pole is hanged to the roof of a cart, and the goal is to swing up the pole by moving the cart properly and try to keep the pole at the top.

The state space $\mathcal{S}$ is two-dimensional and continuous, which consists of the angle $\varphi \in [0, 2\pi]$ and angular velocity $\dot{\varphi} \in [-3\pi, 3\pi]$ of the pole. The action space $\mathcal{A}$ is one-dimensional and continuous, which corresponds to the force applied to the cart (note that we can *not* directly control the pole, but only indirectly through moving the cart). We use the Gaussian policy model for REINFORCE and linear policy model for PGPE, where state $s$ is non-linearly transformed to a feature space via a basis function vector. We use 20 Gaussian kernels with standard deviation $\sigma = 0.5$ as the basis functions, where the kernel centers are distributed over the following grid points: $\{0, \pi/2, \pi, 3\pi/2\} \times \{-3\pi, -3\pi/2, 0, 3\pi/2, 3\pi\}$. The dynamics of the pole (i.e., the update rule of the angle and the angular velocity) is given by

$$\varphi_{t+1} = \varphi_t + \dot{\varphi}_{t+1}\Delta t \quad \text{and} \quad \dot{\varphi}_{t+1} = \dot{\varphi}_t + \frac{9.8\sin(\varphi_t) - \alpha w l \dot{\varphi}_t^2 \sin(2\varphi_t)/2 + \alpha\cos(\varphi_t)a_t}{4l/3 - \alpha w l \cos^2(\varphi_t)}\Delta t,$$

where $\alpha = 1/W + w$ and $a_t$ is the action taken at time $t$. We set the problem parameters as: the mass of the cart $W = 8$[kg], the mass of the pole $w = 2$[kg], and the length of the pole $l = 0.5$[m]. We set the time step $\Delta t$ for the position and velocity updates at $0.01$[s] and action selection at $0.1$[s]. The reward function is defined as $r(s_t, a_t, s_{t+1}) = \cos(\varphi_{t+1})$. That is, the higher the pole is, the more rewards we can obtain. The initial policy is chosen randomly, and the initial-state probability density is set to be uniform. The agent collects $N = 100$ episodic samples with trajectory length $T = 40$. The discount factor is set at $\gamma = 0.9$.

We investigate average returns over 10 trials as the functions of policy-update iterations. The return at each trial is computed over 100 test episodic samples (which are not used for policy learning). The experimental results are plotted in Figure 3, showing that the improvement of both plain REINFORCE and REINFORCE-OB tend to be slow, and all PGPE methods outperformed REINFORCE methods overall. Among the PGPE methods, the proposed PGPE-OB converges faster than others.

## 6 Conclusion

In this paper, we analyzed and improved the stability of the policy gradient method called PGPE (policy gradients with parameter-based exploration). We theoretically showed that, under a mild condition, PGPE provides more stable gradient estimates than the classical REINFORCE method. We also derived the optimal baseline for PGPE, and theoretically showed that PGPE with the optimal baseline is more preferable than REINFORCE with the optimal baseline in terms of the variance of gradient estimates. Finally, we demonstrated the usefulness of PGPE with optimal baseline through experiments.

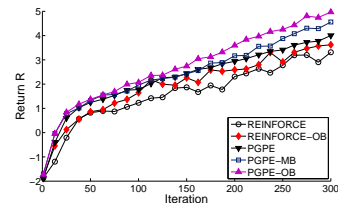

Figure 3: Performance of policy

**Acknowledgments:** TZ and GN were supported by the MEXT scholarship and the GCOE program, HH was supported by the FIRST program, and MS was supported by MEXT KAKENHI 23120004.

# References

[1] N. Abe, P. Melville, C. Pendus, C. K. Reddy, D. L. Jensen, V. P. Thomas, J. J. Bennett, G. F. Anderson, B. R. Cooley, M. Kowalczyk, M. Domick, and T. Gardinier. Optimizing debt collections using constrained reinforcement learning. In *Proceedings of The 16th ACM SGKDD Conference on Knowledge Discovery and Data Mining*, pages 75–84, 2010.

[2] J. Baxter, P. Bartlett, and L. Weaver. Experiments with infinite-horizon, policy-gradient estimation. *Journal of Artificial Intelligence Research*, 15:351–381, 2001.

[3] M. Bugeja. Non-linear swing-up and stabilizing control of an inverted pendulum system. In *Proceedings of IEEE Region 8 EUROCON*, volume 2, pages 437–441, 2003.

[4] P. Dayan and G. E. Hinton. Using expectation-maximization for reinforcement learning. *Neural Computation*, 9(2):271–278, 1997.

[5] E. Greensmith, P. L. Bartlett, and J. Baxter. Variance reduction techniques for gradient estimates in reinforcement learning. *Journal of Machine Learning Research*, 5:1471–1530, 2004.

[6] L. P. Kaelbling, M. L. Littman, and A. W. Moore. Reinforcement learning: A survey. *Journal of Artificial Intelligence Research*, 4:237–285, 1996.

[7] S. Kakade. A natural policy gradient. In T. G. Dietterich, S. Becker, and Z. Ghahramani, editors, *Advances in Neural Information Processing Systems 14*, pages 1531–1538, Cambridge, MA, 2002. MIT Press.

[8] M. G. Lagoudakis and R. Parr. Least-squares policy iteration. *Journal of Machine Learning Research*, 4:1107–1149, 2003.

[9] P. Marbach and J. N. Tsitsiklis. Approximate gradient methods in policy-space optimization of Markov reward processes. *Discrete Event Dynamic Systems*, 13(1-2):111–148, 2004.

[10] J. Peters and S. Schaal. Policy gradient methods for robotics. In *Processing of the IEEE/RSJ International Conferece on Inatelligent Robots and Systems(IROS)*, 2006.

[11] F. Sehnke, C. Osendorfer, T. Rückstiess, A. Graves, J. Peters, and J. Schmidhuber. Policy gradients with parameter-based exploration for control. In *Proceedings of The 18th International Conference on Artificial Neural Networks*, pages 387–396, 2008.

[12] F. Sehnke, C. Osendorfer, T. Rückstiess, A. Graves, J. Peters, and J. Schmidhuber. Parameter-exploring policy gradients. *Neural Networks*, 23(4):551–559, 2010.

[13] R. S. Sutton and G. A. Barto. *Reinforcement Learning: An Introduction*. MIT Press, Cambridge, MA, USA, 1998.

[14] G. Tesauro. TD-gammon, a self-teaching backgammon program, achieves master-level play. *Neural Computation*, 6(2):215–219, 1994.

[15] L. Weaver and J. Baxter. Reinforcement learning from state and temporal differences. Technical report, Department of Computer Science, Australian National University, 1999.

[16] L. Weaver and N. Tao. The optimal reward baseline for gradient-based reinforcement learning. In *Processings of The Seventeeth Conference on Uncertainty in Artificial Intelligence*, pages 538–545, 2001.

[17] J. D. Williams and S. Young. Partially observable Markov decision processes for spoken dialog systems. *Computer Speech and Language*, 21(2):231–422, 2007.

[18] R. J. Williams. Simple statistical gradient-following algorithms for connectionist reinforcement learning. *Machine Learning*, 8:229, 1992.

